# Multiple Relational Embedding

**Roland Memisevic**
Department of Computer Science
University of Toronto
roland@cs.toronto.edu

**Geoffrey Hinton**
Department of Computer Science
University of Toronto
hinton@cs.toronto.edu

## Abstract

We describe a way of using multiple different types of similarity relationship to learn a low-dimensional embedding of a dataset. Our method chooses different, possibly overlapping representations of similarity by individually reweighting the dimensions of a common underlying latent space. When applied to a single similarity relation that is based on Euclidean distances between the input data points, the method reduces to simple dimensionality reduction. If additional information is available about the dataset or about subsets of it, we can use this information to clean up or otherwise improve the embedding. We demonstrate the potential usefulness of this form of semi-supervised dimensionality reduction on some simple examples.

## 1 Introduction

Finding a representation for data in a low-dimensional Euclidean space is useful both for visualization and as prelude to other kinds of data analysis. The common goal underlying the many different methods that accomplish this task (such as ISOMAP [1], LLE [2], stochastic neighbor embedding [3] and others) is to extract the usually small number of factors that are responsible for the variability in the data. In making the underlying factors explicit, these methods help to focus on the kind of variability that is important and provide representations that make it easier to interpret and manipulate the data in reasonable ways.

Most dimensionality reduction methods are unsupervised, so there is no way of guiding the method towards modes of variability that are of particular interest to the user. There is also no way of providing hints when the true underlying factors are too subtle to be discovered by optimizing generic criteria such as maximization of modeled variance in PCA, or preservation of local geometry in LLE. Both these difficulties can be alleviated by allowing the user to provide more information than just the raw data points or a single set of pairwise similarities between data points.

As an example consider images of faces. Nonlinear methods have been shown to find embeddings that nicely reflect the variability in the data caused by variation in face identity, pose, position, or lighting effects. However, it is not possible to tell these methods to extract a particular single factor for the purpose of, say intelligent image manipulation or pose identification, because the extracted factors are intermingled and may be represented simultaneously across all latent space dimensions.

Here, we consider the problem of learning a latent representation for data based on knowl-

edge that is provided by a user in the form of several different similarity relations. Our method, multiple relational embedding (MRE), finds an embedding that uses a single latent data representation, but weights the available latent space dimensions differently to allow the latent space to model the multiple different similarity relations. By labeling a subset of the data according to the kind of variability one is interested in, one can encourage the model to reserve a subset of the latent dimensions for this kind of variability. The model, in turn, returns a "handle" to that latent space in the form of a corresponding learned latent space metric. Like stochastic neighbor embedding, MRE can also be derived as a simplification of Linear Relational Embedding[4].

## 1.1 Related work

The problem of supplementing methods for unsupervised learning with "side-information" in order to influence their solutions is not new and many different approaches have been suggested. [5], for example, describes a way to inform a PCA model by encouraging it to preserve a user-defined grouping structure; [6] consider the problem of extracting exactly two different kinds of factors, which they denote "style" and "content", by using bilinear models; more recently, [7] and [8] took a quite different approach to informing a model. They suggest pre-processing the input data by learning a metric in input space that makes the data respect user defined grouping constraints.

Our approach differs from these and other methods in two basic ways. First, in all the methods mentioned above, the side-information has to be defined in terms of *equivalence constraints*. That is, a user needs to define a grouping structure for the input data by informing the model which data-points belong together. Here, we consider a rather different approach, where the side-information can be encoded in the form of *similarity relations*. This allows arbitrary continuous degrees of freedom to constrain the low-dimensional embeddings. Second, our model can deal with several, possibly conflicting, kinds of side-information. MRE dynamically "allocates" latent space dimensions to model different user-provided similarity relations. So inconsistent relations are modeled in disjoint subspaces, and consistent relations can share dimensions. This scheme of sharing the dimensions of a common latent space is reminiscent of the INDSCAL method [9] that has been popular in the psychometric literature.

A quite different way to extend unsupervised models has recently been introduced by [10] and [11], where the authors propose ways to extract common factors that underlie two or more different datasets, with possibly different dimensionalities. While these methods rely on a supervision signal containing information about correspondences between data-points in different datasets, MRE can be used to *discover* correspondences between different datasets using almost no pre-defined grouping constraints.

## 2 Multiple Relational Embedding

In the following we derive MRE as an extension to stochastic neighbor embedding (SNE). Let $X$ denote the matrix of latent space elements arranged column-wise, and let $\sigma^2$ be some real-valued neighborhood variance or "kernel bandwidth". SNE finds a low-dimensional representation for a set of input data points $y^i(i = 1, \ldots, N)$ by first constructing a similarity matrix $P$ with entries

$$P_{ij} := \frac{\exp(-\frac{1}{\sigma^2}\|y^i - y^j\|^2)}{\sum_k \exp(-\frac{1}{\sigma^2}\|y^i - y^k\|^2)} \qquad (1)$$

and then minimizing (w.r.t. the set of latent space elements $x^i (i = 1, \ldots, N)$) the mismatch between $P$ and the corresponding latent similarity matrix $Q(X)$ defined by

$$Q_{ij}(X) := \frac{\exp(-\|x^i - x^j\|^2)}{\sum_k \exp(-\|x^i - x^k\|^2)}. \tag{2}$$

The (row-) normalization of both matrices arises from SNE's probabilistic formulation in which the $(i, j)^{\text{th}}$ entry of $P$ and $Q$ is interpreted as the probability that the $i^{\text{th}}$ data-point will pick the $j^{\text{th}}$ point as its neighbor (in observable and latent space, respectively). The mismatch is defined as the sum of Kullback-Leibler-divergences between the respective rows [3].

Our goal is to extend SNE so that it learns latent data representations that not only approximate the input space distances well, but also reflect additional characteristics of the input data that one may be interested in. In order to accommodate these additional characteristics, instead of defining a single similarity-matrix that is based on Euclidean distances in data space, we define several matrices $P^c$, $(c = 1, \ldots, C)$, each of which encodes some known type of similarity of the data. Proximity in the Euclidean data-space is typically *one* of the types of similarity that we use, though it can easily be omitted. The additional types of similarity may reflect any information that the user has access to about any subsets of the data provided the information can be expressed as a similarity matrix that is normalized over the relevant subset of the data.

At first sight, a single latent data representation seems to be unsuitable to accommodate the different, and possibly incompatible, properties encoded in a set of $P^c$-matrices. Since our goal, however, is to capture possibly overlapping relations, we do use a single latent space and in addition we define a linear transformation $R^c$ of the latent space for each of the $C$ different similarity-types that we provide as input. Note that this is equivalent to measuring distances in latent space using a different Mahalanobis metric for each $c$ corresponding to the matrix $R^{cT} R^c$.

In order to learn the transformations $R^c$ from the data along with the set of latent representations $X$ we consider the loss function

$$E(X) = \sum_c E^c(X), \tag{3}$$

where we define

$$E^c(X) := \frac{1}{N} \sum_{i,j} P^c_{ij} \log \left( \frac{P^c_{ij}}{Q^c_{ij}} \right) \quad \text{and} \quad Q^c_{ij} := Q_{ij}(R^c X). \tag{4}$$

Note that in the case of $C = 1$, $R^1 = I$ (and fixed) and $P^1$ defined as in Eq. (1) this function simplifies to the standard SNE objective function. One might consider weighting the contribution of each similarity-type using some weighting factor $\lambda^c$. We found that the solutions are rather robust with regard to different sets of $\lambda^c$ and weighted all error contributions equally in our experiments.

As indicated above, here we consider diagonal $R$-matrices only, which simply amounts to using a rescaling factor for each latent space dimension. By allowing each type of similarity to put a different scaling factor on each dimension the model allows similarity relations that "overlap" to share dimensions. Completely unrelated or "orthogonal" relations can be encoded by using disjoint sets of non-zero scaling factors.

The gradient of $E(X)$ w.r.t. a single latent space element $x^l$ takes a similar form to the gradient of the standard SNE objective function and is given by

$$\frac{\partial E(X)}{\partial x^l} = \frac{2}{N} \sum_c \sum_i (P^c_{il} + P^c_{li} - Q^c_{li} - Q^c_{il}) R^{cT} R^c (x^l - x^i), \tag{5}$$

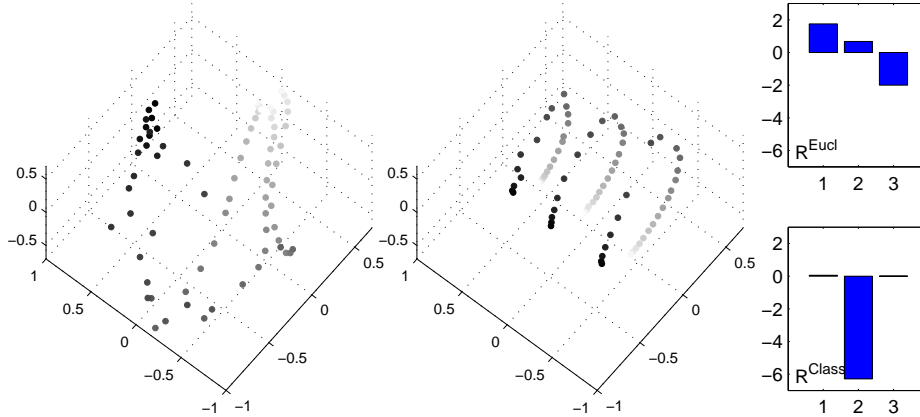

Figure 1: Embedding of images of rotated objects. Left: SNE, right: MRE. Latent representatives are colored on a gray-scale corresponding to angle of rotation in the original images. The rightmost plots show entries on the diagonals of latent space transformations $R^{\text{Eucl}}$ and $R^{\text{Class}}$.

the gradient w.r.t. to a single entry of the diagonal of $R^c$ reads

$$\frac{\partial E(X)}{\partial R_{ll}^c} = \frac{2}{N} R_{ll}^c \left[ \sum_i \sum_j \left( P_{ij}^c - Q_{ij}^c \right) (x_l^i - x_l^j)^2 \right],\qquad(6)$$

where $x_l^i$ denotes the $l^{\text{th}}$ component of the $i^{\text{th}}$ latent representative.

As an illustrative example we ran MRE on a set of images from the Columbia object images library (COIL) [12]. The dataset contains $(128 \times 128)$-dimensional gray-scale images of different objects that vary only by rotation, i.e. by a single degree of freedom. We took three subsets of images depicting toy-cars, where each subset corresponds to one of three different kinds of toy-cars, and embedded the first 30 images of each of these subsets in a three-dimensional space. We used two similarity relations: The first, $P^{\text{Eucl}}$, corresponds to the standard SNE objective; the second, $P^{\text{Class}}$, is defined as a block diagonal matrix that contains homogeneous blocks of size $30 \times 30$ with entries $(\frac{1}{30})$ and models class membership, i.e. we informed the model using the information that images depicting the same object class belong together.

We also ran standard SNE on the same dataset[1]. The results are depicted in figure 1. While SNE's unsupervised objective to preserve Euclidean distances leads to a representation where class-membership is intermingled with variability caused by object rotation (leftmost plot), in the MRE approximation the contribution of class-membership is factored out and represented in a separate dimension (next plot). This is also reflected in the entries on the diagonal of the corresponding R-matrices, depicted in the two right-most plots. $R^{\text{Class}}$ is responsible for representing class membership and can do so using just a single dimension. $R^{\text{Eucl}}$ on the other hand makes use of all dimensions to some degree, reflecting the fact that the overall variability in "pixel-space" depends on class-membership, as well as on other factors (here mainly rotation). Note that with the variability according to class-

membership factored out, the remaining two dimensions capture the rotational degree of freedom very cleanly.

## 2.1 Partial information

In many real world situations there might be side-information available only for a subset of the data-points, because labelling a complete dataset could be too expensive or for other reasons impossible. A partially labelled dataset can in that case still be used to provide a hint about the kind of variability that one is interested in. In general, since the corresponding transformation $R^c$ provides a way to access the latent space that represents the desired similarity-type, a partially labelled dataset can be used to perform a form of supervised feature extraction in which the labelled data is used to specify a kind of feature "by example". It is straightforward to modify the model to deal with partially labelled data. For each type of similarity $c$ that is known to hold for a subset containing $N^c$ examples, the corresponding $P^c$-matrix references only this subset of the complete dataset and is thus an $N^c \times N^c$-matrix. To keep the latent space elements not corresponding to this subset unaffected by this error contribution, we can define for each $c$ an index set $I^c$ containing just the examples referenced by $P^c$ and rewrite the loss for that type of similarity as

$$E^c(X) := \frac{1}{N^c} \sum_{i,j \in I^c} P^c_{ij} \log \left( \frac{P^c_{ij}}{Q^c_{ij}} \right). \tag{7}$$

# 3 Experiments

## 3.1 Learning correspondences between image sets

In extending the experiment described in section 2 we trained MRE to discover correspondences between sets of images, in this case with different dimensionalities. We picked 20 successive images from one object of the COIL dataset described above and 28 images ($112 \times 92$ pixels) depicting a person under different viewing angles taken from the UMIST dataset[13]. We chose this data in order to obtain two sets of images that vary in a "similar" or related way. Note that, because the datasets have different dimensionalities, here it is not possible to define a single relation describing Euclidean distance between all data-points. Instead we constructed two relations $P^{\text{Coil}}$ and $P^{\text{Umist}}$ (for both we used Eq. (1) with $\sigma^2$ set as in the previous experiment), with corresponding index-sets $I^{\text{Coil}}$ and $I^{\text{Umist}}$ containing the indices of the points in each of the two datasets. In addition we constructed one class-membership relation in the same way as before and two identical relations $P^1$ and $P^2$ that take the form of a $2 \times 2$-matrix filled with entries $\frac{1}{2}$. Each of the corresponding index sets $I^1$ and $I^2$ points to two images (one from each dataset) that represent the end points of the rotational degree of freedom, i.e. to the first and the last points if we sort the data according to rotation (see figure 2, left plot). These similarity types are used to make sure that the model properly aligns the representations of the two different datasets. Note that the end points constitute the *only* supervision signal; we did not use any additional information about the alignment of the two datasets.

After training a two-dimensional embedding[2], we randomly picked latent representatives of the COIL images and computed reconstructions of corresponding face images using a kernel smoother (i.e. as a linear combination of the face images with coefficients based on latent space distances). In order to factor out variability corresponding to class membership we first multiplied *all* latent representatives by the inverse of $R^{\text{class}}$. (Note that such a strategy will in general blow up the latent space dimensions that do not represent class membership, as the corresponding entries in $R^{\text{class}}$ may contain very small values. The

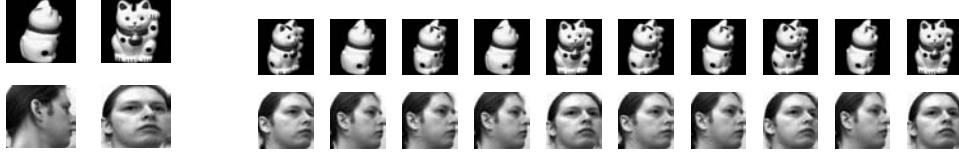

Figure 2: Face reconstructions by alignment. Left: Side-information in form of two image pairs in correspondence. Right: Reconstructions of face images from randomly chosen cat images.

kernel smoother consequently requires a very large kernel bandwidth, with the net effect that the latent representation effectively collapses in the dimensions that correspond to class membership – which is exactly what we want.) The reconstructions, depicted in the right plot of figure 2, show that the model has captured the common mode of variability.

### 3.2   Supervised feature extraction

To investigate the ability of MRE to perform a form of "supervised feature extraction" we used a dataset of synthetic face images that originally appeared in [1]. The face images vary according to pose (two degrees of freedom) and according to the position of a lighting source (one degree of freedom). The corresponding low-dimensional parameters are available for each data-point. We computed an embedding with the goal of obtaining features that explicitly correspond to these different kinds of variability in the data.

We labelled a subset of 100 out of the total of 698 data-points with the three mentioned degrees of freedom in the following way: After standardizing the pose and lighting parameters so that they were centered and had unit variance, we constructed three corresponding similarity matrices ($P^{\mathrm{Pose1}}$, $P^{\mathrm{Pose2}}$, $P^{\mathrm{Lighting}}$) for a randomly chosen subset of 100 points using Eq. (1) and the three low-dimensional parameter sets as input data. In addition we used a fourth similarity relation $P^{\mathrm{Ink}}$, corresponding to overall brightness or "amount of ink", by constructing for each image a corresponding feature equal to the sum of its pixel intensities and then defining the similarity matrix as above. We set the bandwidth parameter $\sigma^2$ to 1.0 for all of these similarity-types[3]. In addition we constructed the standard SNE relation $P^{\mathrm{Eucl}}$ (defined for all data-points) using Eq. (1) with $\sigma^2$ set[4] to 100.

We initialized the model as before and trained for 1000 iterations of 'minimize' to find an embedding in a four-dimensional space. Figure 3 (right plot) shows the learned latent space metrics corresponding to the five similarity-types. Obviously, MRE devotes one dimension to each of the four similarity-types, reflecting the fact that each of them describes a single one-dimensional degree of freedom that is barely correlated with the others. Data-space similarities in contrast are represented using all dimensions. The plots on the left of figure 3 show the embedding of the 598 unlabelled data-points. The top plot shows the embedding in the two dimensions in which the two "pose"-metrics take on their maximal values, the bottom plot shows the dimensions in which the "lighting"- and "ink"-metric take on their maximal values. The plots show that MRE generalizes over unlabeled data: In each dimension the unlabeled data is clearly arranged according to the corresponding similarity type, and is arranged rather randomly with respect to other similarity types. There are a few correlations, in particular between the first pose- and the "ink"-parameter, that are inherent in the dataset, i.e. the data does not vary entirely independently with respect to these parameters. These correlations are also reflected in the slightly overlapping latent

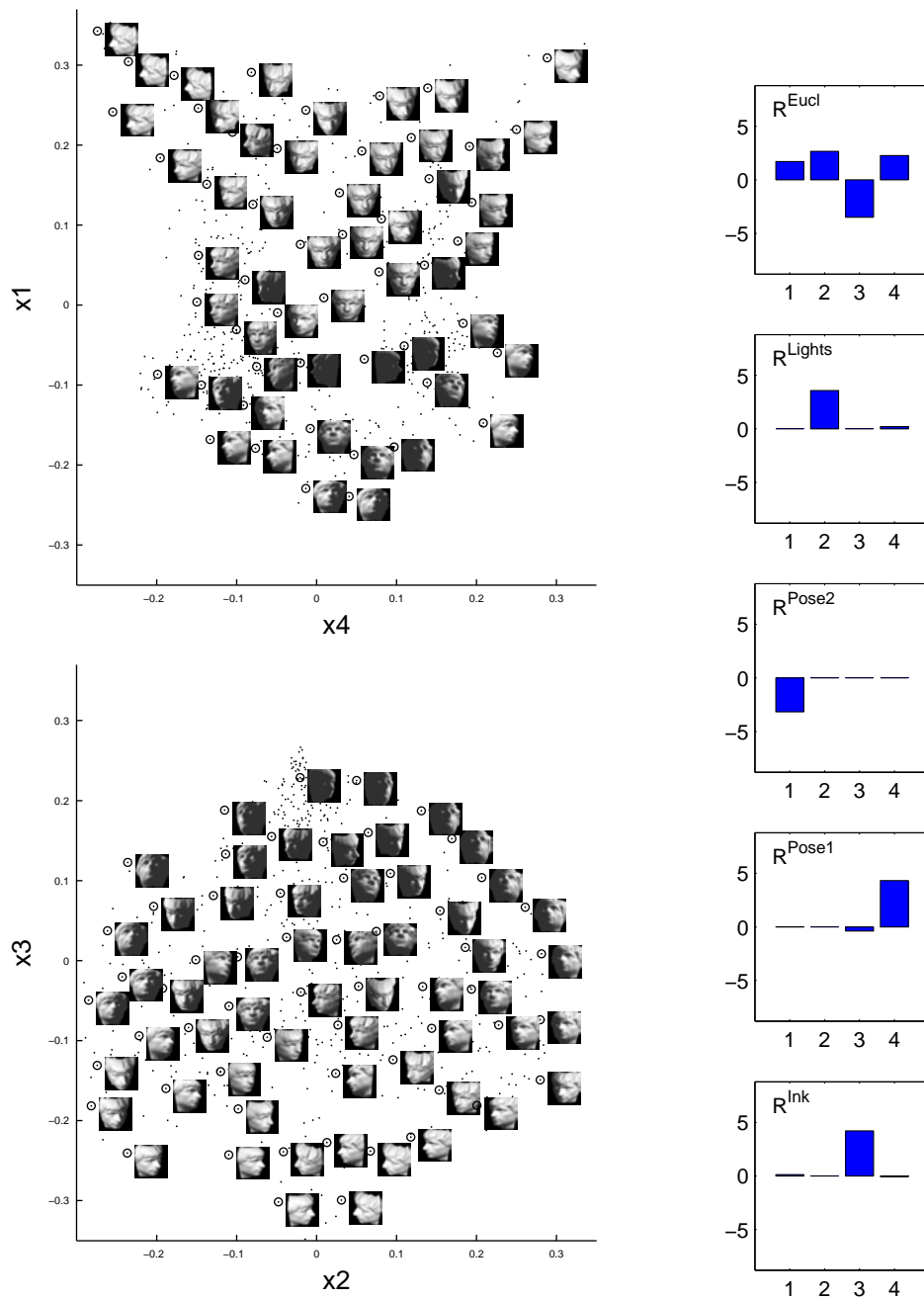

Figure 3: Left: Embedding of faces images that were not informed about their low-dimensional parameters. For a randomly chosen subset of these (marked with a circle), the original images are shown next to their latent representatives. Right: Entries on the diagonals of five latent space transformations.

space weight sets. MRE gets the pose-embedding wrong for a few very dark images that are apparently too far away in the data space to be associated with the correct labeled data-points.

## 4 Conclusions

We introduced a way to embed data in a low-dimensional space using a set of similarity relations. Our experiments indicate that the informed feature extraction that this method facilitates will be most useful in cases where conventional dimensionality reduction methods fail because of their completely unsupervised nature. Although we derived our approach as an extension to SNE, it should be straightforward to apply the same idea to other dimensionality reduction methods.

**Acknowledgements:** Roland Memisevic is supported by a Government of Canada Award. Geoffrey Hinton is a fellow of CIAR and holds a CRC chair. This research was also supported by grants from NSERC and CFI.

## Footnotes

[1]For training we set $\sigma^2$ manually to $5 \cdot 10^7$ for both SNE and MRE and initialized all entries in $X$ and the diagonals of all $R^c$ with small normally distributed values. In all experiments we minimized the loss function defined in Eq. (3) using Carl Rasmussens' matlab function "minimize" for 200 iterations (simple gradient descent worked equally well, but was much slower).

[2]Training was done using 500 iterations with a setup as in the previous experiment.

[3]This is certainly not an optimal choice, but we found the solutions to be rather robust against changes in the bandwidth, and this value worked fine.

[4]See previous footnote.

## References

[1] Joshua B. Tenenbaum, Vin de Silva, and John C. Langford. A global geometric framework for nonlinear dimensionality reduction. *Science*, pages 2319–2323, 2000.

[2] S.T. Roweis and L. K. Saul. Nonlinear dimensionality reduction by locally linear embedding. *Science*, 290, 2000.

[3] Geoffrey Hinton and Sam Roweis. Stochastic neighbor embedding. In *Advances in Neural Information Processing Systems 15*, pages 833–840. MIT Press, 2003.

[4] A. Paccanaro and G. E. Hinton. Learning hierarchical structures with linear relational embedding. In *Advances in Neural Information Processing Systems 14*, Cambridge, MA, 2002. MIT Press.

[5] David Cohn. Informed projections. In *Advances in Neural Information Processing Systems 15*, pages 849–856. MIT Press, 2003.

[6] Joshua B. Tenenbaum and William T. Freeman. Separating style and content with bilinear models. *Neural Computation*, 12(6):1247–1283, 2000.

[7] Eric P. Xing, Andrew Y. Ng, Michael I. Jordan, and Stuart Russell. Distance metric learning with application to clustering with side-information. In *Advances in Neural Information Processing Systems 15*, pages 505–512. MIT Press, Cambridge, MA, 2003.

[8] Michinari Momma Tijl De Bie and Nello Cristianini. Efficiently learning the metric using side-information. In *Proc. of the 14th International Conference on Algorithmic Learning Theory*, 2003.

[9] J. Douglas Carroll and Jih-Jie Chang. Analysis of individual differences in multidimensional scaling via an n-way generalization of "eckart-young" decomposition. *Psychometrika*, 35(3), 1970.

[10] J. H. Ham, D. D. Lee, and L. K. Saul. Learning high dimensional correspondences from low dimensional manifolds. In *In Proceedings of the ICML 2003 Workshop on The Continuum from Labeled to Unlabeled Data in Machine Learning and Data Mining*, pages 34–41, Washington, D.C., 2003.

[11] Jakob J. Verbeek, Sam T. Roweis, and Nikos Vlassis. Non-linear cca and pca by alignment of local models. In *Advances in Neural Information Processing Systems 16*. MIT Press, Cambridge, MA, 2004.

[12] S. A. Nene, S. K. Nayar, and H. Murase. Columbia object image library (coil-20). Technical report, 1996.

[13] Daniel B Graham and Nigel M Allinson. Characterizing virtual eigensignatures for general purpose face recognition. 163, 1998.
